# Variance Reduction in Monte-Carlo Tree Search

**Joel Veness**
University of Alberta
veness@cs.ualberta.ca

**Marc Lanctot**
University of Alberta
lanctot@cs.ualberta.ca

**Michael Bowling**
University of Alberta
bowling@cs.ualberta.ca

## Abstract

Monte-Carlo Tree Search (MCTS) has proven to be a powerful, generic planning technique for decision-making in single-agent and adversarial environments. The stochastic nature of the Monte-Carlo simulations introduces errors in the value estimates, both in terms of bias and variance. Whilst reducing bias (typically through the addition of domain knowledge) has been studied in the MCTS literature, comparatively little effort has focused on reducing variance. This is somewhat surprising, since variance reduction techniques are a well-studied area in classical statistics. In this paper, we examine the application of some standard techniques for variance reduction in MCTS, including common random numbers, antithetic variates and control variates. We demonstrate how these techniques can be applied to MCTS and explore their efficacy on three different stochastic, single-agent settings: Pig, Can't Stop and Dominion.

## 1   Introduction

Monte-Carlo Tree Search (MCTS) has become a popular approach for decision making in large domains. The fundamental idea is to iteratively construct a search tree, whose internal nodes contain value estimates, by using Monte-Carlo simulations. These value estimates are used to direct the growth of the search tree and to estimate the value under the optimal policy from each internal node. This general approach [6] has been successfully adapted to a variety of challenging problem settings, including Markov Decision Processes, Partially Observable Markov Decision Processes, Real-Time Strategy games, Computer Go and General Game Playing [15, 22, 2, 9, 12, 10].

Due to its popularity, considerable effort has been made to improve the efficiency of Monte-Carlo Tree Search. Noteworthy enhancements include the addition of domain knowledge [12, 13], parallelization [7], Rapid Action Value Estimation (RAVE) [11], automated parameter tuning [8] and rollout policy optimization [21]. Somewhat surprisingly however, the application of classical variance reduction techniques to MCTS has remained unexplored. In this paper we survey some common variance reduction ideas and show how they can be used to improve the efficiency of MCTS.

For our investigation, we studied three stochastic games: Pig [16], Can't Stop [19] and Dominion [24]. We found that substantial increases in performance can be obtained by using the appropriate combination of variance reduction techniques. To the best of our knowledge, our work constitutes the first investigation of classical variance reduction techniques in the context of MCTS. By showing some examples of these techniques working in practice, as well as discussing the issues involved in their application, this paper aims to bring this useful set of techniques to the attention of the wider MCTS community.

## 2   Background

We begin with a short overview of Markov Decision Processes and online planning using Monte-Carlo Tree Search.

## 2.1 Markov Decision Processes

A Markov Decision Process (MDP) is a popular formalism [4, 23] for modeling sequential decision making problems. Although more general setups exist, it will be sufficient to limit our attention to the case of finite MDPs. Formally, a finite MDP is a triplet $(\mathcal{S}, \mathcal{A}, \mathcal{P}_0)$, where $\mathcal{S}$ is a finite, non-empty set of states, $\mathcal{A}$ is a finite, non-empty set of actions and $\mathcal{P}_0$ is the transition probability kernel that assigns to each state-action pair $(s, a) \in \mathcal{S} \times \mathcal{A}$ a probability measure over $\mathcal{S} \times \mathbb{R}$ that we denote by $\mathcal{P}_0(\cdot \mid s, a)$. $\mathcal{S}$ and $\mathcal{A}$ are known as the *state space* and *action space* respectively. Without loss of generality, we assume that the state always contains the current time index $t \in \mathbb{N}$. The transition probability kernel gives rise to the *state transition kernel* $\mathcal{P}(s, a, s') := \mathcal{P}_0(\{s'\} \times \mathbb{R} \mid s, a)$, which gives the probability of transitioning from state $s$ to state $s'$ if action $a$ is taken in $s$. An agent's behavior can be described by a *policy* that defines, for each state $s \in \mathcal{S}$, a probability measure over $\mathcal{A}$ denoted by $\pi(\cdot \mid s)$. At each time $t$, the agent communicates an action $A_t \sim \pi(\cdot \mid S_t)$ to the system in state $S_t \in \mathcal{S}$. The system then responds with a state-reward pair $(S_{t+1}, R_{t+1}) \sim \mathcal{P}_0(\cdot \mid S_t, A_t)$, where $S_{t+1} \in \mathcal{S}$ and $R_{t+1} \in \mathbb{R}$. We will assume that each reward lies within $[r_{\min}, r_{\max}] \subset \mathbb{R}$ and that the system executes for only a finite number of steps $n \in \mathbb{N}$ so that $t \leq n$. Given a sequence of random variables $A_t, S_{t+1}, R_{t+1}, \ldots, A_{n-1}, S_n, R_n$ describing the execution of the system up to time $n$ from a state $s_t$, the *return* from $s_t$ is defined as $X_{s_t} := \sum_{i=t+1}^{n} R_i$. The return $X_{s_t, a_t}$ with respect to a state-action pair $(s_t, a_t) \in \mathcal{S} \times \mathcal{A}$ is defined similarly, with the added constraint that $A_t = a_t$. An *optimal policy*, denoted by $\pi^*$, is a policy that maximizes the expected return $\mathbb{E}[X_{s_t}]$ for all states $s_t \in \mathcal{S}$. A deterministic optimal policy always exists for this class of MDPs.

## 2.2 Online Monte-Carlo Planning in MDPs

If the state space is small, an optimal action can be computed offline for each state using techniques such as exhaustive Expectimax Search [18] or Q-Learning [23]. Unfortunately, state spaces too large for these approaches are regularly encountered in practice. One way to deal with this is to use *online planning*. This involves repeatedly using search to compute an approximation to the optimal action from the current state. This effectively amortizes the planning effort across multiple time steps, and implicitly focuses the approximation effort on the relevant parts of the state space.

A popular way to construct an online planning algorithm is to use a depth-limited version of an exhaustive search technique (such as Expectimax Search) in conjunction with iterative deepening [18]. Although this approach works well in domains with limited stochasticity, it scales poorly in highly stochastic MDPs. This is because of the exhaustive enumeration of all possible successor states at chance nodes. This enumeration severely limits the maximum search depth that can be obtained given reasonable time constraints. Depth-limited exhaustive search is generally outperformed by Monte-Carlo planning techniques in these situations.

A canonical example of online Monte-Carlo planning is 1-ply rollout-based planning [3]. It combines a *default policy* $\pi$ with a one-ply lookahead search. At each time $t < n$, given a starting state $s_t$, for each $a_t \in \mathcal{A}$ and with $t < i < n$, $\mathbb{E}[X_{s_t, a_t} \mid A_i \sim \pi(\cdot \mid S_i)]$ is estimated by generating trajectories $S_{t+1}, R_{t+1}, \ldots, A_{n-1}, S_n, R_n$ of agent-system interaction. From these trajectories, sample means $\bar{X}_{s_t, a_t}$ are computed for all $a_t \in \mathcal{A}$. The agent then selects the action $A_t := \operatorname{argmax}_{a_t \in \mathcal{A}} \bar{X}_{s_t, a}$, and observes the system response $(S_{t+1}, R_{t+1})$. This process is then repeated until time $n$. Under some mild assumptions, this technique is provably superior to executing the default policy [3]. One of the main advantages of rollout based planning compared with exhaustive depth-limited search is that a much larger search horizon can be used. The disadvantage however is that if $\pi$ is suboptimal, then $\mathbb{E}[X_{s_t, a} \mid A_i \sim \pi(\cdot \mid S_i)] < \mathbb{E}[X_{s_t, a} \mid A_i \sim \pi^*(\cdot \mid S_i)]$ for at least one state-action pair $(s_t, a) \in \mathcal{S} \times \mathcal{A}$, which implies that at least some value estimates constructed by 1-ply rollout-based planning are biased. This can lead to mistakes which cannot be corrected through additional sampling. The bias can be reduced by incorporating more knowledge into the default policy, however this can be both difficult and time consuming.

Monte-Carlo Tree Search algorithms improve on this procedure, by providing a means to construct asymptotically consistent estimates of the return under the optimal policy from simulation trajectories. The UCT algorithm [15] in particular has been shown to work well in practice. Like rollout-based planning, it uses a default policy to generate trajectories of agent-system interaction. However now the construction of a search tree is also interleaved within this process, with nodes corresponding to states and edge corresponding to a state-action pairs. Initially, the search tree consists of a

single node, which represents the current state $s_t$ at time $t$. One or more *simulations* are then performed. We will use $\mathcal{T}_m \subset \mathcal{S}$ to denote the set of states contained within the search tree after $m \in \mathbb{N}$ simulations. Associated with each state-action pair $(s, a) \in \mathcal{S} \times \mathcal{A}$ is an estimate $\bar{X}_{s,a}^m$ of the return under the optimal policy and a count $T_{s,a}^m \in \mathbb{N}$ representing the number of times this state-action pair has been visited after $m$ simulations, with $T_{s,a}^0 := 0$ and $\bar{X}_{s,a}^0 := 0$.

Each simulation can be broken down into four phases, *selection*, *expansion*, *rollout* and *backup*. Selection involves traversing a path from the root node to a leaf node in the following manner: for each non-leaf, internal node representing some state $s$ on this path, the UCB [1] criterion is applied to select an action until a leaf node corresponding to state $s_l$ is reached. If $\mathcal{U}(\mathcal{B}_s)$ denotes the uniform distribution over the set of unexplored actions $\mathcal{B}_s^m := \{a \in \mathcal{A} : T_{s,a}^m = 0\}$, and $T_s^m := \sum_{a \in \mathcal{A}} T_{s,a}^m$, UCB at state $s$ selects

$$A_s^{m+1} := \underset{a \in \mathcal{A}}{\operatorname{argmax}} \; \bar{X}_{s,a}^m + c\sqrt{\log(T_s^m)/T_{s,a}^m}, \tag{1}$$

if $|\mathcal{B}_s^m| = \emptyset$, or $A_s^{m+1} \sim \mathcal{U}(\mathcal{B}_s^m)$ otherwise. The ratio of exploration to exploitation is controlled by the positive constant $c \in \mathbb{R}$. In the case of more than one maximizing action, ties are broken uniformly at random. Provided $s_l$ is non-terminal, the expansion phase is then executed, by selecting an action $A_l \sim \pi(\cdot \,|\, s_l)$, observing a successor state $S_{l+1} = s_{l+1}$, and then adding a node to the search tree so that $\mathcal{T}_{m+1} = \mathcal{T}_m \cup \{s_{l+1}\}$. Higher values of $c$ increase the level of exploration, which in turn leads to more shallow and symmetric tree growth. The rollout phase is then invoked, which for $l < i < n$, executes actions $A_i \sim \pi(\cdot \,|\, S_i)$. At this point, a complete agent-system execution trajectory $(a_t, s_{t+1}, r_{t+1}, \ldots, a_{n-1}, s_n, r_n)$ from $s_t$ has been realized. The backup phase then assigns, for $t \le k < n$,

$$\bar{X}_{s_k,a_k}^{m+1} \leftarrow \bar{X}_{s_k,a_k}^m + \tfrac{1}{T_{s_k,a_k}^m + 1}\left(\sum_{i=t+1}^n r_i - \bar{X}_{s_k,a_k}^m\right), \qquad T_{s_k,a_k}^{m+1} \leftarrow T_{s_k,a_k}^m + 1,$$

to each $(s_k, a_k) \in \mathcal{T}_{m+1}$ occurring on the realized trajectory. Notice that for all $(s, a) \in \mathcal{S} \times \mathcal{A}$, the value estimate $\bar{X}_{s,a}^m$ corresponds to the average return of the realized simulation trajectories passing through state-action pair $(s, a)$. After the desired number of simulations $k$ has been performed in state $s_t$, the action with the highest expected return $a_t := \operatorname{argmax}_{a \in \mathcal{A}} \bar{X}_{s_t,a}^k$ is selected. With an appropriate [15] value of $c$, as $m \to \infty$, the value estimates converge to the expected return under the optimal policy. However, due to the stochastic nature of the UCT algorithm, each value estimate $\bar{X}_{s,a}^m$ is subject to error, in terms of both bias and variance, for finite $m$. While previous work (see Section 1) has focused on improving these estimates by reducing bias, little attention has been given to improvements via variance reduction. The next section describes how the accuracy of UCT's value estimates can be improved by adapting classical variance reduction techniques to MCTS.

## 3 Variance Reduction in MCTS

This section describes how three variance reduction techniques — control variates, common random numbers and antithetic variates — can be applied to the UCT algorithm. Each subsection begins with a short overview of each variance reduction technique, followed by a description of how UCT can be modified to efficiently incorporate it. Whilst we restrict our attention to planning in MDPs using the UCT algorithm, the ideas and techniques we present are quite general. For example, similar modifications could be made to the Sparse Sampling [14] or AMS [5] algorithms for planning in MDPs, or to the POMCP algorithm [22] for planning in POMDPs. In what follows, given an independent and identically distributed sample $(X_1, X_2, \ldots X_n)$, the sample mean is denoted by $\bar{X} := \frac{1}{n}\sum_{i=1}^n X_i$. Provided $\mathbb{E}[X]$ exists, $\bar{X}$ is an unbiased estimator of $\mathbb{E}[X]$ with variance $\sqrt{n^{-1}\mathbb{V}\mathrm{ar}[X]}$.

### 3.1 Control Variates

An improved estimate of $\mathbb{E}[X]$ can be constructed if we have access to an additional statistic $Y$ that is correlated with $X$, provided that $\mu_Y := \mathbb{E}[Y]$ exists and is known. To see this, note that if $Z := X + c(Y - \mathbb{E}[Y])$, then $\bar{Z}$ is an unbiased estimator of $\mathbb{E}[X]$, for any $c \in \mathbb{R}$. $Y$ is called the *control variate*. One can show that $\mathbb{V}\mathrm{ar}[Z]$ is minimised for $c^* := -\mathbb{C}\mathrm{ov}[X, Y]/\mathbb{V}\mathrm{ar}[Y]$. Given a sample $(X_1, Y_1), (X_2, Y_2), \ldots, (X_n, Y_n)$ and setting $c = c^*$, the control variate enhanced estimator

$$\bar{X}_{cv} := \frac{1}{n}\sum_{i=1}^n \left[X_i + c^*(Y_i - \mu_Y)\right] \tag{2}$$

is obtained, with variance

$$\mathbb{V}\mathrm{ar}[\bar{X}_{cv}] = \frac{1}{n}\left(\mathbb{V}\mathrm{ar}[X] - \frac{\mathbb{C}\mathrm{ov}[X,Y]^2}{\mathbb{V}\mathrm{ar}[Y]}\right).$$

Thus the total variance reduction is dependent on the strength of correlation between $X$ and $Y$. For the optimal value of $c$, the variance reduction obtained by using $Z$ in place of $X$ is $100 \times \mathbb{C}\mathrm{orr}[X,Y]^2$ percent. In practice, both $\mathbb{V}\mathrm{ar}[Y]$ and $\mathbb{C}\mathrm{ov}[X,Y]$ are unknown and need to be estimated from data. One solution is to use the plug-in estimator $C_n := -\widehat{\mathbb{C}\mathrm{ov}}[X,Y]/\widehat{\mathbb{V}\mathrm{ar}}(Y)$, where $\widehat{\mathbb{C}\mathrm{ov}}[\cdot,\cdot]$ and $\widehat{\mathbb{V}\mathrm{ar}}(\cdot)$ denote the sample covariance and sample variance respectively. This estimate can be constructed offline using an independent sample or be estimated online. Although replacing $c^*$ with an online estimate of $C_n$ in Equation 2 introduces bias, this modified estimator is still consistent [17]. Thus online estimation is a reasonable choice for large $n$; we revisit the issue of small $n$ later. Note that $\bar{X}_{cv}$ can be efficiently computed with respect to $C_n$ by maintaining $\bar{X}$ and $\bar{Y}$ online, since $\bar{X}_{cv} = \bar{X} + C_n(\bar{Y} - \mu_Y)$.

**Application to UCT.** Control variates can be applied recursively, by redefining the return $X_{s,a}$ for every state-action pair $(s,a) \in \mathcal{S} \times \mathcal{A}$ to

$$Z_{s,a} := X_{s,a} + c_{s,a}\left(Y_{s,a} - \mathbb{E}[Y_{s,a}]\right), \tag{3}$$

provided $\mathbb{E}\left[Y_{s,a}\right]$ exists and is known for all $(s,a) \in \mathcal{S} \times \mathcal{A}$, and $Y_{s,a}$ is a function of the random variables $A_t, S_{t+1}, R_{t+1}, \ldots, A_{n-1}, S_n, R_n$ that describe the complete execution of the system after action $a$ is performed in state $s$. Notice that a separate control variate will be introduced for each state-action pair. Furthermore, as $\mathbb{E}\left[Z_{s_t,a_t} \mid A_i \sim \pi(\cdot \mid S_i)\right] = \mathbb{E}\left[X_{s_t,a_t} \mid A_i \sim \pi(\cdot \mid S_i)\right]$, for all policies $\pi$, for all $(s_t, a_t) \in \mathcal{S} \times \mathcal{A}$ and for all $t < i < n$, the inductive argument [15] used to establish the asymptotic consistency of UCT still applies when control variates are introduced in this fashion.

Finding appropriate control variates whose expectations are known in advance can prove difficult. This situation is further complicated in UCT where we seek a set of control variates $\{Y_{s,a}\}$ for all $(s,a) \in \mathcal{S} \times \mathcal{A}$. Drawing inspiration from advantage sum estimators [25], we now provide a general class of control variates designed for application in UCT. Given a realization of a random simulation trajectory $S_t = s_t, A_t = a_t, S_{t+1} = s_{t+1}, A_{t+1} = a_{t+1}, \ldots, S_n = s_n$, consider control variates of the form

$$Y_{s_t,a_t} := \sum_{i=t}^{n-1} \mathbb{I}[b(S_{i+1})] - \mathbb{P}[b(S_{i+1}) \mid S_i = s_i, A_i = a_i], \tag{4}$$

where $b : \mathcal{S} \to \{\mathrm{true}, \mathrm{false}\}$ denotes a boolean function of state and $\mathbb{I}$ denotes the binary indicator function. In this case, the expectation

$$\mathbb{E}[Y_{s_t,a_t}] = \sum_{i=t}^{n-1}\left(\mathbb{E}\left[\mathbb{I}\left[b(S_{i+1})\right] \mid S_i = s_i, A_i = a_i\right] - \mathbb{P}\left[b(S_{i+1}) \mid S_i = s_i, A_i = a_i\right]\right) = 0,$$

for all $(s_t, a_t) \in \mathcal{S} \times \mathcal{A}$. Thus, using control variates of this form simplifies the task to specifying a state property that is strongly correlated with the return, such that $\mathbb{P}[b(S_{i+1}) \mid S_i = s_i, A_i = a_i]$ is known for all $(s_i, a_i) \in (\mathcal{S}, \mathcal{A})$, for all $t \le i < n$. This considerably reduces the effort required to find an appropriate set of control variates for UCT.

## 3.2 Common Random Numbers

Consider comparing the expectation of $\mathbb{E}[Y]$ to $\mathbb{E}[Z]$, where both $Y := g(X)$ and $Z := h(X)$ are functions of a common random variable $X$. This can be framed as estimating the value of $\delta_{Y,Z}$, where $\delta_{Y,Z} := \mathbb{E}[g(X)] - \mathbb{E}[h(X)]$. If the expectations $\mathbb{E}[g(X)]$ and $\mathbb{E}[h(X)]$ were estimated from two independent samples $\mathbf{X}_1$ and $\mathbf{X}_2$, the estimator $\hat{g}(\mathbf{X}_1) - \hat{h}(\mathbf{X}_2)$ would be obtained, with variance $\mathbb{V}\mathrm{ar}[\hat{g}(\mathbf{X}_1) - \hat{h}(\mathbf{X}_2)] = \mathbb{V}\mathrm{ar}[\hat{g}(\mathbf{X}_1)] + \mathbb{V}\mathrm{ar}[\hat{h}(\mathbf{X}_2)]$. Note that no covariance term appears since $\mathbf{X}_1$ and $\mathbf{X}_2$ are independent samples. The technique of *common random numbers* suggests setting $\mathbf{X}_1 = \mathbf{X}_2$ if $\mathbb{C}\mathrm{ov}(\hat{g}(\mathbf{X}_1), \hat{h}(\mathbf{X}_2))$ is positive. This gives the estimator $\hat{\delta}_{Y,Z}(\mathbf{X}_1) := \hat{g}(\mathbf{X}_1) - \hat{h}(\mathbf{X}_1)$, with variance $\mathbb{V}\mathrm{ar}[\hat{g}(\mathbf{X}_1)] + \mathbb{V}\mathrm{ar}[\hat{h}(\mathbf{X}_1)] - 2\mathbb{C}\mathrm{ov}[\hat{g}(\mathbf{X}_1), \hat{h}(\mathbf{X}_1)]$, which is an improvement whenever $\mathbb{C}\mathrm{ov}[\hat{g}(\mathbf{X}_1), \hat{h}(\mathbf{X}_1)]$ is positive. This technique cannot be applied indiscriminately however, since a variance increase will result if the estimates are negatively correlated.

**Application to UCT.** Rather than directly reducing the variance of the individual return estimates, common random numbers can instead be applied to reduce the variance of the estimated differences

in return $\bar{X}^m_{s,a} - \bar{X}^m_{s,a'}$, for each pair of distinct actions $a, a' \in \mathcal{A}$ in a state $s$. This has the benefit of reducing the effect of variance in both determining the action $a_t := \operatorname{argmax}_{a \in \mathcal{A}} \bar{X}^m_{s,a}$ selected by UCT in state $s_t$ and the actions $\operatorname{argmax}_{a \in \mathcal{A}} \bar{X}^m_{s,a} + c\sqrt{\log(T^m_s)/T^m_{s,a}}$ selected by UCB as the search tree is constructed.

As each estimate $\bar{X}^m_{s,a}$ is a function of realized simulation trajectories originating from state-action pair $(s, a)$, a carefully chosen subset of the stochastic events determining the realized state transitions now needs to be shared across future trajectories originating from $s$ so that $\mathbb{C}\mathrm{ov}[\bar{X}^m_{s,a}, \bar{X}^m_{s,a'}]$ is positive for all $m \in \mathbb{N}$ and for all distinct pairs of actions $a, a' \in \mathcal{A}$. Our approach is to use the same chance outcomes to determine the trajectories originating from state-action pairs $(s, a)$ and $(s, a')$ if $T^i_{s,a} = T^j_{s,a'}$, for any $a, a' \in \mathcal{A}$ and $i, j \in \mathbb{N}$. This can be implemented by using $T^m_{s,a}$ to index into a list of stored stochastic outcomes $E_s$ defined for each state $s$. By only adding a new outcome to $E_s$ when $T_{s,a}$ exceeds the number of elements in $E_s$, the list of common chance outcomes can be efficiently generated online. This idea can be applied recursively, provided that the shared chance events from the current state do not conflict with those defined at any possible ancestor state.

### 3.3 Antithetic Variates

Consider estimating $\mathbb{E}[X]$ with $\hat{h}(\mathbf{X}, \mathbf{Y}) := \frac{1}{2}\hat{h}_1(\mathbf{X}) + \hat{h}_2(\mathbf{Y})$, the average of two unbiased estimates $\hat{h}_1(\mathbf{X})$ and $\hat{h}_2(\mathbf{Y})$, computed from two identically distributed samples $\mathbf{X} = (X_1, X_2, \ldots, X_n)$ and $\mathbf{Y} = (Y_1, Y_2, \ldots, Y_n)$. The variance of $\hat{h}(\mathbf{X}, \mathbf{Y})$ is

$$\frac{1}{4}(\mathbb{V}\mathrm{ar}[\hat{h}_1(\mathbf{X})] + \mathbb{V}\mathrm{ar}[\hat{h}_2(\mathbf{Y})]) + \frac{1}{2}\mathbb{C}\mathrm{ov}[\hat{h}_1(\mathbf{X}), \hat{h}_2(\mathbf{Y})]. \tag{5}$$

The method of *antithetic variates* exploits this identity, by deliberately introducing a negative correlation between $\hat{h}_1(\mathbf{X})$ and $\hat{h}_2(\mathbf{Y})$. The usual way to do this is to construct $\mathbf{X}$ and $\mathbf{Y}$ from pairs of sample points $(X_i, Y_i)$ such that $\mathbb{C}\mathrm{ov}[h_1(X_i), h_2(Y_i)] < 0$ for all $i \leq n$. So that $\hat{h}_2(\mathbf{Y})$ remains an unbiased estimate of $\mathbb{E}[X]$, care needs to be taken when making $\mathbf{Y}$ depend on $\mathbf{X}$.

**Application to UCT.** Like the technique of common random numbers, antithetic variates can be applied to UCT by modifying the way simulation trajectories are sampled. Whenever a node representing $(s_i, a_i) \in \mathcal{S} \times \mathcal{A}$ is visited during the backup phase of UCT, the realized trajectory $s_{i+1}, r_{i+1}, a_{i+1}, \ldots, s_n, r_n$ from $(s_i, a_i)$ is now stored in memory if $T^m_{s_i,a_i} \bmod 2 \equiv 0$. The next time this node is visited during the selection phase, the previous trajectory is used to predetermine one or more antithetic events that will (partially) drive subsequent state transitions for the current simulation trajectory. After this, the memory used to store the previous simulation trajectory is reclaimed. This technique can be applied to all state-action pairs inside the tree, provided that the antithetic events determined by any state-action pair do not overlap with the antithetic events defined by any possible ancestor.

## 4 Empirical Results

This section begins with a description of our test domains, and how our various variance reduction ideas can be applied to them. We then investigate the performance of UCT when enhanced with various combinations of these techniques.

### 4.1 Test Domains

Pig is a turn-based jeopardy dice game that can be played with one or more players [20]. Players roll two dice each turn and keep a turn total. At each decision point, they have two actions, roll and stop. If they decide to stop, they add their turn total to their total score. Normally, dice rolls add to the players turn total, with the following exceptions: if a single ⬚ is rolled the turn total will be reset and the turn ended; if a ⬚⬚ is rolled then the players turn will end along with their *total* score being reset to 0. These possibilities make the game highly stochastic.

Can't Stop is a dice game where the goal is to obtain three complete columns by reaching the highest level in each of the 2-12 columns [19]. This done by repeatedly rolling 4 dice and playing zero or more pairing combinations. Once a pairing combination is played, a marker is placed on the associated column and moved upwards. Only three distinct columns can be used during any

given turn. If the dice are rolled and no legal pairing combination can be made, the player loses all of the progress made towards completing columns on this turn. After rolling and making a legal pairing, a player can chose to lock in their progress by ending their turn. A key component of the game involves correctly assessing the risk associated with not being able to make a legal dice pairing given the current board configuration.

Dominion is a popular turn-based, deck-building card game [24]. It involves acquiring cards by spending the money cards in your current deck. Bought cards have certain effects that allow you to buy more cards, get more money, draw more cards, and earn victory points. The goal is to get as many victory points as possible.

In all cases, we used solitaire variants of the games where the aim is to maximize the number of points given a fixed number of turns. All of our domains can be represented as finite MDPs. The game of Pig contains approximately $2.4 \times 10^6$ states. Can't Stop and Dominion are significantly more challenging, containing in excess of $10^{24}$ and $10^{30}$ states respectively.

## 4.2 Application of Variance Reduction Techniques

We now describe the application of each technique to the games of Pig, Can't Stop and Dominion.

**Control Variates.** The control variates used for all domains were of the form specified by Equation 4 in Section 3.1. In Pig, we used a boolean function that returned true if we had just performed the roll action and obtained at least one ⚀. This control variate has an intuitive interpretation, since we would expect the return from a single trajectory to be an underestimate if it contained more rolls with a ⚀ than expected, and an overestimate if it contained less rolls with a ⚀ than expected. In Can't Stop, we used similarly inspired boolean function that returned true if we could not make a legal pairing from our most recent roll of the 4 dice. In Dominion, we used a boolean function that returned whether we had just played an action that let us randomly draw a hand with 8 or more money to spend. This is a significant occurrence, as 8 money is needed to buy a Province, the highest scoring card in the game. Strong play invariably requires purchasing as many Provinces as possible.

We used a mixture of online and offline estimation to determine the values of $c_{s,a}$ to use in Equation 3. When $T_{s,a}^m \geq 50$, the online estimate $-\widehat{\mathbb{C}\text{ov}}[X_{s,a}, Y_{s,a}]/\widehat{\mathbb{V}\text{ar}}[Y_{s,a}]$ was used. If $T_{s,a}^m < 50$, the constants $6.0$, $6.0$ and $-0.7$ were used for Pig, Can't Stop and Dominion respectively. These constants were obtained by computing offline estimates of $-\widehat{\mathbb{C}\text{ov}}[X_{s,a}, Y_{s,a}]/\widehat{\mathbb{V}\text{ar}}[Y_{s,a}]$ across a representative sample of game situations. This combination gave better performance than either scheme in isolation.

**Common Random Numbers.** To apply the ideas in Section 3.2, we need to specify the future chance events to be shared across all of the trajectories originating from each state. Since a player's final score in Pig is strongly dependent on their dice rolls, it is natural to consider sharing one or more future dice roll outcomes. By exploiting the property in Pig that each roll event is independent of the current state, our implementation shares a batch of roll outcomes large enough to drive a complete simulation trajectory. So that these chance events don't conflict, we limited the sharing of roll events to just the root node. A similar technique was used in Can't Stop. We found this scheme to be superior to sharing a smaller number of future roll outcomes and applying the ideas in Section 3.2 recursively. In Dominion, stochasticity is caused by drawing cards from the top of a deck that is periodically shuffled. Here we implemented common random numbers by recursively sharing pre-shuffled deck configurations across the actions at each state. The motivation for this kind of sharing is that it should reduce the chance of one action appearing better than another simply because of "luckier" shuffles.

**Antithetic Variates.** To apply the ideas in Section 3.3, we need to describe how the antithetic events are constructed from previous simulation trajectories. In Pig, a negative correlation between the returns of pairs of simulation trajectories can be induced by forcing the roll outcomes in the second trajectory to oppose those occurring in the first trajectory. Exploiting the property that the relative worth of each pair of dice outcomes is independent of state, a list of antithetic roll outcomes can be constructed by mapping each individual roll outcome in the first trajectory to its antithetic partner. For example, a lucky roll of ⚅⚅ was paired with the unlucky roll of ⚀⚀. A similar idea is used in Can't Stop, however the situation is more complicated, since the relative worth of each

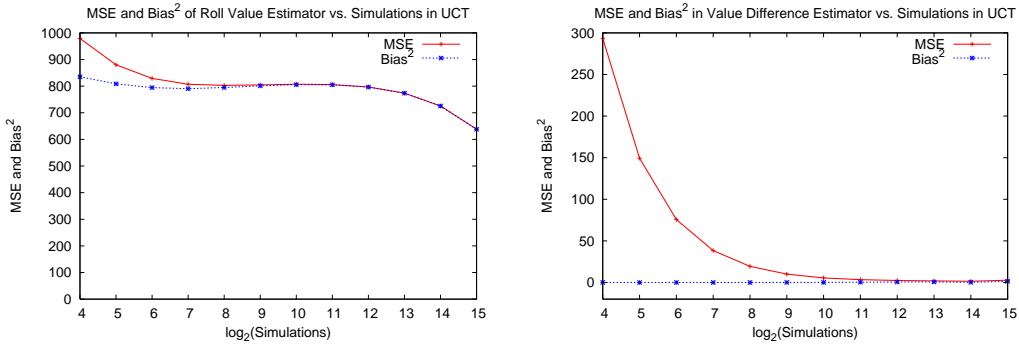

*Figure 1:* The estimated variance of the value estimates for the Roll action and estimated differences between actions on turn 1 in Pig.

chance event varies from state to state. Our solution was to develop a state-dependent heuristic ranking function, which would assign an index between 0 and 1295 to the $6^4$ distinct chance events for a given state. Chance events that are favorable in the current state are assigned low indexes, while unfavorable events are assigned high index values. When simulating a non-antithetic trajectory, the ranking for each chance event is recorded. Later when the antithetic trajectory needs to be simulated, the previously recorded rank indexes are used to compute the relevant antithetic event for the current state. This approach can be applied in a wide variety of domains where the stochastic outcomes can be ordered by how "lucky" they are e.g., suppliers' price fluctuations, rare catastrophic events, or higher than average click-through-rates. For Dominion, a number of antithetic mappings were tried, but none provided any substantial reduction in variance. The complexity of how cards can be played to draw more cards from one's deck makes a good or bad shuffle intricately dependent on the exact composition of cards in one's deck, of which there are intractably many possibilities with no obvious symmetries.

## 4.3 Experimental Setup

Each variance reduction technique is evaluated in combination with the UCT algorithm, with varying levels of search effort. In Pig, the default (rollout) policy plays the roll and stop actions with probability $0.8$ and $0.2$ respectively. In Can't Stop, the default policy will end the turn if a column has just been finished, otherwise it will choose to re-roll with probability $0.85$. In Dominion, the default policy incorporates some simple domain knowledge that favors obtaining higher cost cards and avoiding redundant actions. The UCB constant $c$ in Equation 1 was set to $100.0$ for both Pig and Dominion and $5500.0$ for Can't Stop.

## 4.4 Evaluation

We performed two sets of experiments. The first is used to gain a deeper understanding of the role of bias and variance in UCT. The next set of results is used to assess the overall performance of UCT when augmented with our variance reduction techniques.

**Bias versus Variance.** When assessing the quality of an estimator using *mean squared error* (MSE), it is well known that the estimation error can be decomposed into two terms, bias and variance. Therefore, when assessing the potential impact of variance reduction, it is important to know just how much of the estimation error is caused by variance as opposed to bias. Since the game of Pig has $\approx 2.4 \times 10^6$ states, we can solve it offline using Expectimax Search. This allows us to compute the expected return $\mathbb{E}[X_{s_1} \,|\, \pi^*]$ of the optimal action (roll) at the starting state $s_1$. We use this value to compute both the bias-squared and variance component of the MSE for the estimated return of the roll action at $s_1$ when using UCT without variance reduction. This is shown in the leftmost graph of Figure 1. It seems that the dominating term in the MSE is the bias-squared. This is misleading however, since the absolute error is not the only factor in determining which action is selected by UCT. More important instead is the *difference* between the estimated returns for each action, since UCT ultimately ends up choosing the action with the largest estimated return. As Pig has just two actions, we can also compute the MSE of the estimated difference in return between rolling and stopping using UCT without variance reduction. This is shown by the rightmost graph

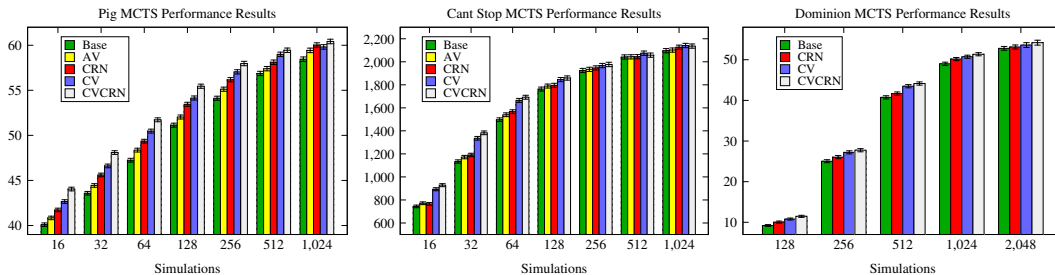

*Figure 2:* Performance Results for Pig, Can't Stop, and Dominion with 95% confidence intervals shown. Values on the vertical axis of each graph represent the average score.

in Figure 1. Here we see that variance is the dominating component (the bias is within $\pm 2$) when the number of simulations is less than $1024$. The role of bias and variance will of course vary from domain to domain, but this result suggests that variance reduction may play an important role when trying to determine the best action.

**Search Performance.** Figure 2 shows the results of our variance reduction methods on Pig, Can't Stop and Dominion. Each data point for Pig, Can't Stop and Dominion is obtained by averaging the scores obtained across $50,000$, $10,000$ and $10,000$ games respectively. Such a large number of games is needed to obtain statistically significant results due to the highly stochastic nature of each domain. 95% confidence intervals are shown for each data point. In Pig, the best approach consistently outperforms the base version of UCT, even when given twice the number of simulations. In Can't Stop, the best approach gave a performance increase roughly equivalent to using base UCT with 50-60% more simulations. The results also show a clear benefit to using variance reduction techniques in the challenging game of Dominion. Here the best combination of variance reduction techniques leads to an improvement roughly equivalent to using 25-40% more simulations. The use of antithetic variates in both Pig and Can't Stop gave a measurable increase in performance, however the technique was less effective than either control variates or common random numbers. Control variates was particularly helpful across all domains, and even more effective when combined with common random numbers.

## 5  Discussion

Although our UCT modifications are designed to be lightweight, some additional overhead is unavoidable. Common random numbers and antithetic variates increase the space complexity of UCT by a multiplicative constant. Control variates typically increase the time complexity of each value backup by a constant. These factors need to be taken into consideration when evaluating the benefits of variance reduction for a particular domain. Note that surprising results are possible; for example, if generating the underlying chance events is expensive, using common random numbers or antithetic variates can even *reduce* the computational cost of each simulation. Ultimately, the effectiveness of variance reduction in MCTS is both domain and implementation specific. That said, we would expect our techniques to be useful in many situations, especially in noisy domains or if each simulation is computationally expensive. In our experiments, the overhead of every technique was dominated by the cost of simulating to the end of the game.

## 6  Conclusion

This paper describes how control variates, common random numbers and antithetic variates can be used to improve the performance of Monte-Carlo Tree Search by reducing variance. Our main contribution is to describe how the UCT algorithm can be modified to efficiently incorporate these techniques in practice. In particular, we provide a general approach that significantly reduces the effort needed recursively apply control variates. Using these methods, we demonstrated substantial performance improvements on the highly stochastic games of Pig, Can't Stop and Dominion. Our work should be of particular interest to those using Monte-Carlo planning in highly stochastic or resource limited settings.

# References

[1] Peter Auer. Using confidence bounds for exploitation-exploration trade-offs. *JMLR*, 3:397–422, 2002.

[2] Radha-Krishna Balla and Alan Fern. UCT for Tactical Assault Planning in Real-Time Strategy Games. In *IJCAI*, pages 40–45, 2009.

[3] Dimitri P. Bertsekas and David A. Castanon. Rollout algorithms for stochastic scheduling problems. *Journal of Heuristics*, 5(1):89–108, 1999.

[4] Dimitri P. Bertsekas and John N. Tsitsiklis. *Neuro-Dynamic Programming*. Athena Scientific, 1st edition, 1996.

[5] Hyeong S. Chang, Michael C. Fu, Jiaqiao Hu, and Steven I. Marcus. An Adaptive Sampling Algorithm for Solving Markov Decision Processes. *Operations Research*, 53(1):126–139, January 2005.

[6] Guillaume Chaslot, Sander Bakkes, Istvan Szita, and Pieter Spronck. Monte-Carlo Tree Search: A New Framework for Game AI. In *Fourth Artificial Intelligence and Interactive Digital Entertainment Conference (AIIDE 2008)*, 2008.

[7] Guillaume M. Chaslot, Mark H. Winands, and H. Jaap Herik. Parallel Monte-Carlo Tree Search. In *Proceedings of the 6th International Conference on Computers and Games*, pages 60–71, Berlin, Heidelberg, 2008. Springer-Verlag.

[8] Guillaume M.J-B. Chaslot, Mark H.M. Winands, Istvan Szita, and H. Jaap van den Herik. Cross-entropy for Monte-Carlo Tree Search. *ICGA*, 31(3):145–156, 2008.

[9] Rémi Coulom. Efficient selectivity and backup operators in Monte-Carlo tree search. In *Proceedings Computers and Games 2006*. Springer-Verlag, 2006.

[10] Hilmar Finnsson and Yngvi Bjornsson. Simulation-based Approach to General Game Playing. In *Twenty-Third AAAI Conference on Artificial Intelligence (AAAI 2008)*, pages 259–264, 2008.

[11] S. Gelly and D. Silver. Combining online and offline learning in UCT. In *Proceedings of the 17th International Conference on Machine Learning*, pages 273–280, 2007.

[12] Sylvain Gelly and Yizao Wang. Exploration exploitation in Go: UCT for Monte-Carlo Go. In *NIPS Workshop on On-line trading of Exploration and Exploitation*, 2006.

[13] Sylvain Gelly, Yizao Wang, Rémi Munos, and Olivier Teytaud. Modification of UCT with patterns in Monte-Carlo Go. Technical Report 6062, INRIA, France, November 2006.

[14] Michael J. Kearns, Yishay Mansour, and Andrew Y. Ng. A sparse sampling algorithm for near-optimal planning in large Markov Decision Processes. In *IJCAI*, pages 1324–1331, 1999.

[15] Levente Kocsis and Csaba Szepesvári. Bandit based Monte-Carlo planning. In *ECML*, pages 282–293, 2006.

[16] Todd W. Neller and Clifton G.M. Presser. Practical play of the dice game pig. *Undergraduate Mathematics and Its Applications*, 26(4):443–458, 2010.

[17] Barry L. Nelson. Control variate remedies. *Operations Research*, 38(6):pp. 974–992, 1990.

[18] Stuart Russell and Peter Norvig. *Artificial Intelligence: A Modern Approach*. Prentice-Hall, Englewood Cliffs, NJ, 2nd edition edition, 2003.

[19] Sid Sackson. Can't Stop. *Ravensburger*, 1980.

[20] John Scarne. Scarne on dice. *Harrisburg, PA: Military Service Publishing Co*, 1945.

[21] David Silver and Gerald Tesauro. Monte-Carlo simulation balancing. In *ICML*, page 119, 2009.

[22] David Silver and Joel Veness. Monte-Carlo Planning in Large POMDPs. In *Advances in Neural Information Processing Systems 23*, pages 2164–2172, 2010.

[23] Csaba Szepesvári. Reinforcement learning algorithms for MDPs, 2009.

[24] Donald X. Vaccarino. Dominion. *Rio Grande Games*, 2008.

[25] Martha White and Michael Bowling. Learning a value analysis tool for agent evaluation. In *Proceedings of the Twenty-First International Joint Conference on Artificial Intelligence (IJCAI)*, pages 1976–1981, 2009.

